# Learning To Play the Game of Chess

**Sebastian Thrun**
University of Bonn
Department of Computer Science III
Römerstr. 164, D-53117 Bonn, Germany
E-mail: thrun@carbon.informatik.uni-bonn.de

## Abstract

This paper presents NeuroChess, a program which learns to play chess from the final outcome of games. NeuroChess learns chess board evaluation functions, represented by artificial neural networks. It integrates inductive neural network learning, temporal differencing, and a variant of explanation-based learning. Performance results illustrate some of the strengths and weaknesses of this approach.

## 1 Introduction

Throughout the last decades, the game of chess has been a major testbed for research on artificial intelligence and computer science. Most of today's chess programs rely on intensive search to generate moves. To evaluate boards, fast evaluation functions are employed which are usually carefully designed by hand, sometimes augmented by automatic parameter tuning methods [1]. Building a chess machine that learns to play solely from the final outcome of games (win/loss/draw) is a challenging open problem in AI.

In this paper, we are interested in learning to play chess from the final outcome of games. One of the earliest approaches, which learned solely by playing itself, is Samuel's famous checker player program [10]. His approach employed *temporal difference learning* (in short: TD) [14], which is a technique for recursively learning an evaluation function. Recently, Tesauro reported the successful application of TD to the game of Backgammon, using artificial neural network representations [16]. While his TD-Gammon approach plays grandmaster-level backgammon, recent attempts to reproduce these results in the context of Go [12] and chess have been less successful. For example, Schäfer [11] reports a system just like Tesauro's TD-Gammon, applied to learning to play certain chess endgames. Gherrity [6] presented a similar system which he applied to entire chess games. Both approaches learn purely inductively from the final outcome of games. Tadepalli [15] applied a lazy version of explanation-based learning [5, 7] to endgames in chess. His approach learns from the final outcome, too, but unlike the inductive neural network approaches listed above it learns analytically, by analyzing and generalizing experiences in terms of chess-specific knowledge.

The level of play reported for all these approaches is still below the level of GNU-Chess, a publicly available chess tool which has frequently been used as a benchmark. This illustrates the hardness of the problem of learning to play chess from the final outcome of games.

This paper presents NeuroChess, a program that learns to play chess from the final outcome of games. The central learning mechanisms is the explanation-based neural network (EBNN) algorithm [9, 8]. Like Tesauro's TD-Gammon approach, NeuroChess constructs a neural network evaluation function for chess boards using TD. In addition, a neural network version of explanation-based learning is employed, which analyzes games in terms of a previously learned neural network chess model. This paper describes the NeuroChess approach, discusses several training issues in the domain of chess, and presents results which elucidate some of its strengths and weaknesses.

## 2  Temporal Difference Learning in the Domain of Chess

Temporal difference learning (TD) [14] comprises a family of approaches to prediction in cases where the event to be predicted may be delayed by an unknown number of time steps. In the context of game playing, TD methods have frequently been applied to learn functions which predict the final outcome of games. Such functions are used as board evaluation functions.

The goal of TD(0), a basic variant of TD which is currently employed in the NeuroChess approach, is to find an evaluation function, $V$, which ranks chess boards according to their goodness: If the board $s$ is more likely to be a winning board than the board $s'$, then $V(s) > V(s')$. To learn such a function, TD transforms entire chess games, denoted by a sequence of chess boards $s_0, s_1, s_2, \ldots, s_{t_{\text{final}}}$, into training patterns for $V$. The TD(0) learning rule works in the following way. Assume without loss of generality we are learning white's evaluation function. Then the target values for the *final board* is given by

$$V^{\text{target}}(s_{t_{\text{final}}}) = \begin{cases} 1, & \text{if } s_{t_{\text{final}}} \text{ is a win for white} \\ 0, & \text{if } s_{t_{\text{final}}} \text{ is a draw} \\ -1, & \text{if } s_{t_{\text{final}}} \text{ is a loss for white} \end{cases} \tag{1}$$

and the targets for the intermediate chess boards $s_0, s_1, s_2, \ldots, s_{t_{\text{final}}-2}$ are given by

$$V^{\text{target}}(s_t) = \gamma \cdot V(s_{t+2}) \tag{2}$$

This update rule constructs $V$ recursively. At the end of the game, $V$ evaluates the final outcome of the game (Eq. (1)). In between, when the assignment of $V$-values is less obvious, $V$ is trained based on the evaluation two half-moves later (Eq. (2)). The constant $\gamma$ (with $0 \leq \gamma \leq 1$) is a so-called *discount factor*. It decays $V$ exponentially in time and hence favors early over late success. Notice that in NeuroChess $V$ is represented by an artificial neural network, which is trained to fit the target values $V^{\text{target}}$ obtained via Eqs. (1) and (2) (*cf.* [6, 11, 12, 16]).

## 3  Explanation-Based Neural Network Learning

In a domain as complex as chess, pure inductive learning techniques, such as neural network Back-Propagation, suffer from enormous training times. To illustrate why, consider the situation of a *knight fork*, in which the opponent's knight attacks our queen and king simultaneously. Suppose in order to save our king we have to move it, and hence sacrifice our queen. To learn the badness of a knight fork, NeuroChess has to discover that certain board features (like the position of the queen relative to the knight) are important, whereas

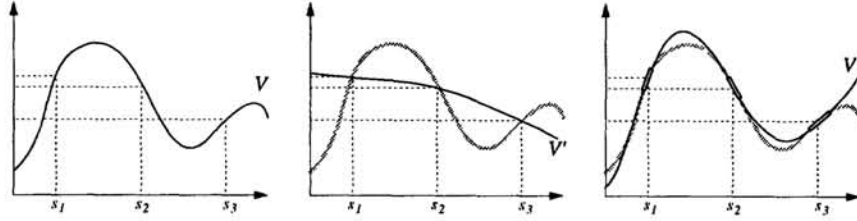

Figure 1: **Fitting values and slopes in EBNN:** Let $V$ be the target function for which three examples $\langle s_1, V(s_1) \rangle$, $\langle s_2, V(s_2) \rangle$, and $\langle s_3, V(s_3) \rangle$ are known. Based on these points the learner might generate the hypothesis $V'$. If the slopes $\frac{\partial V(s_1)}{\partial s_1}$, $\frac{\partial V}{\partial (}s_2)\partial s_2$, and $\frac{\partial V(s_3)}{\partial s_3}$ are also known, the learner can do much better: $V''$.

---

others (like the number of weak pawns) are not. Purely inductive learning algorithms such as Back-propagation figure out the relevance of individual features by observing statistical correlations in the training data. Hence, quite a few versions of a knight fork have to be experienced in order to generalize accurately. In a domain as complex as chess, such an approach might require unreasonably large amounts of training data.

Explanation-based methods (EBL) [5, 7, 15] generalize more accurately from less training data. They rely instead on the availability of domain knowledge, which they use for explaining and generalizing training examples. For example, in the explanation of a knight fork, EBL methods employ knowledge about the game of chess to figure out that the position of the queen is relevant, whereas the number of weak pawns is not. Most current approaches to EBL require that the domain knowledge be represented by a set of symbolic rules. Since NeuroChess relies on neural network representations, it employs a neural network version of EBL, called *explanation-based neural network learning (EBNN)* [9]. In the context of chess, EBNN works in the following way: The domain-specific knowledge is represented by a separate neural network, called the *chess model $M$*. $M$ maps arbitrary chess boards $s_t$ to the corresponding expected board $s_{t+2}$ two half-moves later. It is trained prior to learning $V$, using a large database of grand-master chess games. Once trained, $M$ captures important knowledge about temporal dependencies of chess board features in high-quality chess play.

EBNN exploits $M$ to bias the board evaluation function $V$. It does this by extracting slope constraints for the evaluation function $V$ at all non-final boards, *i.e.*, all boards for which $V$ is updated by Eq. (2). Let

$$\frac{\partial V^{\text{target}}(s_t)}{\partial s_t} \quad \text{with} \quad t \in \{0, 1, 2, \dots, t_{\text{final}} - 2\} \tag{3}$$

denote the target slope of $V$ at $s_t$, which, because $V^{\text{target}}(s_t)$ is set to $\gamma V(s_{t+2})$ according Eq. (2), can be rewritten as

$$\frac{\partial V^{\text{target}}(s_t)}{\partial s_t} = \gamma \cdot \frac{\partial V(s_{t+2})}{\partial s_{t+2}} \cdot \frac{\partial s_{t+2}}{\partial s_t} \tag{4}$$

using the chain rule of differentiation. The rightmost term in Eq. (4) measures how infinitesimal small changes of the chess board $s_t$ influence the chess board $s_{t+2}$. It can be approximated by the chess model $M$:

$$\frac{\partial V^{\text{target}}(s_t)}{\partial s_t} \approx \gamma \cdot \frac{\partial V(s_{t+2})}{\partial s_{t+2}} \cdot \frac{\partial M(s_t)}{\partial s_t} \tag{5}$$

The right expression is only an approximation to the left side, because $M$ is a trained neural

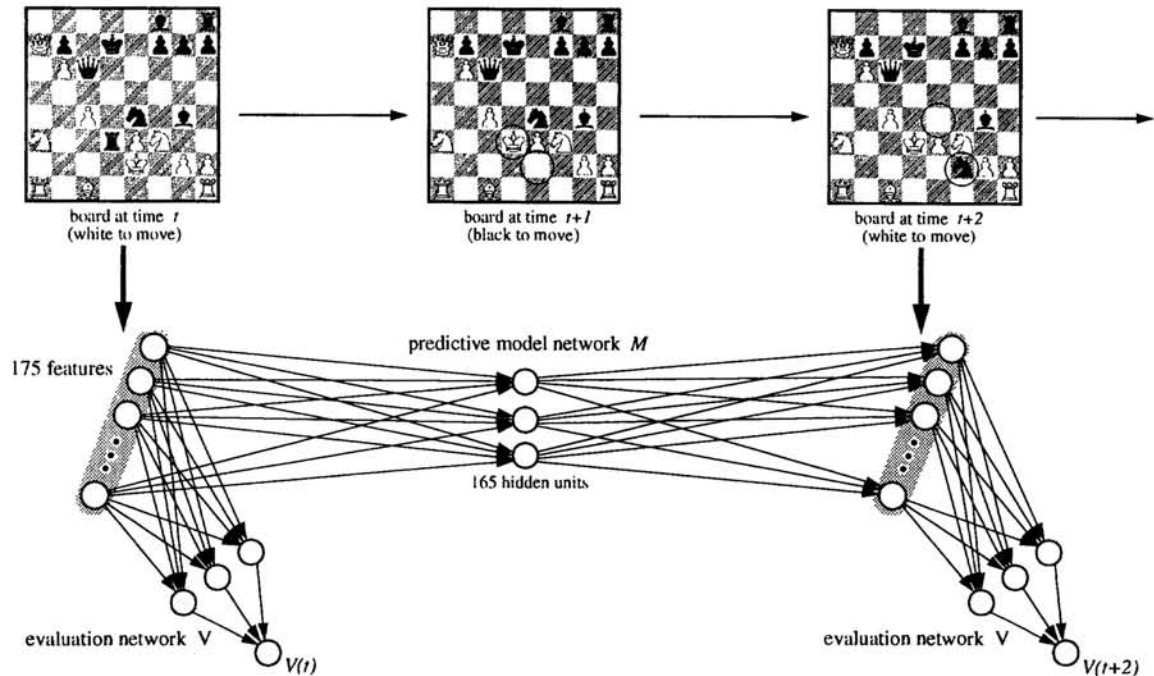

Figure 2: **Learning an evaluation function in NeuroChess.** Boards are mapped into a high-dimensional *feature vector*, which forms the input for both the evaluation network $V$ and the chess model $M$. The evaluation network is trained by Back-propagation and the TD(0) procedure. Both networks are employed for analyzing training example in order to derive target slopes for $V$.

network and thus its first derivative might be erroneous. Notice that both expressions on the right hand side of Eq. (5) are derivatives of neural network functions, which are easy to compute since neural networks are differentiable.

The result of Eq. (5) is an estimate of the slope of the target function $V$ at $s_t$. This slope adds important shape information to the target values constructed via Eq. (2). As depicted in Fig. 1, functions can be fit more accurately if in addition to target values the slopes of these values are known. Hence, instead of just fitting the target values $V^{\text{target}}(s_t)$, NeuroChess also fits these target slopes. This is done using the Tangent-Prop algorithm [13].

The complete NeuroChess learning architecture is depicted in Fig. 2. The target slopes provide a first-order approximation to the relevance of each chess board feature in the goodness of a board position. They can be interpreted as biasing the network $V$ based on chess-specific domain knowledge, embodied in $M$. For the relation of EBNN and EBL and the accommodation of inaccurate slopes in EBNN see [8].

## 4   Training Issues

In this section we will briefly discuss some training issues that are essential for learning good evaluation functions in the domain of chess. This list of points has mainly been produced through practical experience with the NeuroChess and related TD approaches. It illustrates the importance of a careful design of the input representation, the sampling rule and the

parameter setting in a domain as complex as chess.

**Sampling.** The vast majority of chess boards are, loosely speaking, not interesting. If, for example, the opponent leads by more than a queen and a rook, one is most likely to loose. Without an appropriate sampling method there is the danger that the learner spends most of its time learning from uninteresting examples. Therefore, NeuroChess interleaves self-play and expert play for guiding the sampling process. More specifically, after presenting a random number of expert moves generated from a large database of grand-master games, NeuroChess completes the game by playing itself. This sampling mechanism has been found to be of major importance to learn a good evaluation function in a reasonable amount of time.

**Quiescence.** In the domain of chess certain boards are harder to evaluate than others. For example, in the middle of an ongoing material exchange, evaluation functions often fail to produce a good assessment. Thus, most chess programs search selectively. A common criterion for determining the depth of search is called *quiescence*. This criterion basically detects material threats and deepens the search correspondingly. NeuroChess' search engine does the same. Consequently, the evaluation function $V$ is only trained using quiescent boards.

**Smoothness.** Obviously, using the raw, canonical board description as input representation is a poor choice. This is because small changes on the board can cause a huge difference in value, contrasting the smooth nature of neural network representations. Therefore, NeuroChess maps chess board descriptions into a set of board features. These features were carefully designed by hand.

**Discounting.** The variable $\gamma$ in Eq. (2) allows to discount values in time. Discounting has frequently been used to bound otherwise infinite sums of pay-off. One might be inclined to think that in the game of chess no discounting is needed, as values are bounded by definition. Indeed, without discounting the evaluation function predicts the *probability* for winning—in the ideal case. In practice, however, random disturbations of the evaluation function can seriously hurt learning, for reasons given in [4, 17]. Empirically we found that learning failed completely when no discount factor was used. Currently, NeuroChess uses $\gamma = 0.98$.

**Learning rate**. TD approaches minimize a Bellman equation [2]. In the NeuroChess domain, a close-to-optimal approximation of the Bellman equation is the constant function $V(s) \equiv 0$. This function violates the Bellman equation only at the end of games (Eq. (1)), which is rare if complete games are considered. To prevent this, we amplified the learning rate for final values by a factor of 20, which was experimentally found to produce sufficiently non-constant evaluation functions.

**Software architecture**. Training is performed completely asynchronously on up to 20 workstations simultaneously. One of the workstations acts as a weight server, keeping track of the most recent weights and biases of the evaluation network. The other workstations can dynamically establish links to the weight server and contribute to the process of weight refinement. The main process also monitors the state of all other workstations and restarts processes when necessary. Training examples are stored in local ring buffers (1000 items per workstation).

## 5  Results

In this section we will present results obtained with the NeuroChess architecture. Prior to learning an evaluation function, the model $M$ (175 input, 165 hidden, and 175 output units) is trained using a database of 120,000 expert games. NeuroChess then learns an evaluation

| | | | | | |
|---|---|---|---|---|---|
| 1. e2e3 b8c6 | 16. b2b4 a5a4 | 31. a3f8 f2e4 | 46. d1c2 b8h2 | 61. e4f5 h3g4 | 65. a8e8 e6d7 |
| 2. d1f3 c6e5 | 17. b5c6 a4c6 | 32. c3b2 h8f8 | 47. c2c3 f6b6 | 62. f5f6 h6h5 | 66. e8e7 d7d8 |
| 3. f3d5 d7d6 | 18. g1f3 d8d6 | 33. a4d7 f3f5 | 48. e7e4 g6h6 | 63. b7b8q g4f5 | 67. f4c7 |
| 4. f1b5 c7c6 | 19. d4a7 f5g4 | 34. d7b7 f5e5 | 49. d4f5 h6g5 | 64. b8f4 f5e6 | |
| 5. b5a4 g8f6 | 20. c2c4 c8d7 | 35. b2c1 f8e8 | 50. e4e7 g5g4 | | |
| 6. d5d4 c8f5 | 21. b4b5 c6c7 | 36. b7d5 e5h2 | 51. f5h6 g7h6 | **final board** | |
| 7. f2f4 e5d7 | 22. d2d3 d6d3 | 37. a1a7 e8e6 | 52. e7d7 g4h5 | | |
| 8. e1e2 d8a5 | 23. b5b6 c7c6 | 38. d5d8 f6g6 | 53. d7d1 h5h4 | | |
| 9. a4b3 d7c5 | 24. e2d3 e4f2 | 39. b6b7 e6d6 | 54. d1d4 h4h3 | | |
| 10. b1a3 c5b3 | 25. d3c3 g4f3 | 40. d8a5 d6c6 | 55. d4b6 h2e5 | | |
| 11. a2b3 e7e5 | 26. g2f3 f2h1 | 41. a5b4 h2b8 | 56. b6d4 e5e6 | | |
| 12. f4e5 f6e4 | 27. c1b2 c6f3 | 42. a7a8 e4c3 | 57. c3d2 e6f5 | | |
| 13. e5d6 e8c8 | 28. a7a4 d7e7 | 43. c2d4 c6f6 | 58. e3e4 f5g5 | | |
| 14. b3b4 a5a6 | 29. a3c2 h1f2 | 44. b4e7 c3a2 | 59. d4e3 g5e3 | | |
| 15. b4b5 a6a5 | 30. b2a3 e7f6 | 45. c1d1 a2c3 | 60. d2e3 f7f5 | | |

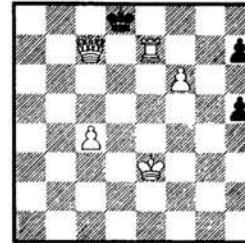

Figure 3: **NeuroChess against GNU-Chess.** NeuroChess plays white. Parameters: Both players searched to depth 3, which could be extended by quiescence search to at most 11. The evaluation network had no hidden units. Approximately 90% of the training boards were sampled from expert play.

network $V$ (175 input units, 0 to 80 hidden units, and one output units). To evaluate the level of play, NeuroChess plays against GNU-Chess in regular time intervals. Both players employ the same search mechanism which is adopted from GNU-Chess. Thus far, experiments lasted for 2 days to 2 weeks on 1 to 20 SUN Sparc Stations.

A typical game is depicted in Fig. 3. This game has been chosen because it illustrates both the strengths and the shortcomings of the NeuroChess approach. The opening of NeuroChess is rather weak. In the first three moves NeuroChess moves its queen to the center of the board.[1] NeuroChess then escapes an attack on its queen in move 4, gets an early pawn advantage in move 12, attacks black's queen pertinaciously through moves 15 to 23, and successfully exchanges a rook. In move 33, it captures a strategically important pawn, which, after chasing black's king for a while and sacrificing a knight for no apparent reason, finally leads to a new queen (move 63). Four moves later black is mate. This game is prototypical. As can be seen from this and various other games, NeuroChess has learned successfully to protect its material, to trade material, and to protect its king. It has not learned, however, to open a game in a coordinated way, and it also frequently fails to play short endgames even if it has a material advantage (this is due to the short planning horizon). Most importantly, it still plays incredibly poor openings, which are often responsible for a draw or a loss. Poor openings do not surprise, however, as TD propagates values from the end of a game to the beginning.

Table 1 shows a performance comparison of NeuroChess versus GNU-Chess, with and without the explanation-based learning strategy. This table illustrates that NeuroChess wins approximately 13% of all games against GNU-Chess, if both use the same search engine. It

| # of games | GNU depth 2, NeuroChess depth 2 | | GNU depth 4, NeuroChess depth 2 | |
|---|---|---|---|---|
| | Back-propagation | EBNN | Back-propagation | EBNN |
| 100 | 1 | 0 | 0 | 0 |
| 200 | 6 | 2 | 0 | 0 |
| 500 | 35 | 13 | 1 | 0 |
| 1000 | 73 | 85 | 2 | 1 |
| 1500 | 130 | 135 | 3 | 3 |
| 2000 | 190 | 215 | 3 | 8 |
| 2400 | 239 | 316 | 3 | 11 |

Table 1: Performance of NeuroChess vs. GNU-Chess during training. The numbers show the total number of games won against GNU-Chess using the same number of games for testing as for training. This table also shows the importance of the explanation-based learning strategy in EBNN. Parameters: both learners used the original GNU-Chess features, the evaluation network had 80 hidden units and search was cut at depth 2, or 4, respectively (no quiescence extensions).

also illustrates the utility of explanation-based learning in chess.

## 6  Discussion

This paper presents NeuroChess, an approach for learning to play chess from the final outcomes of games. NeuroChess integrates TD, inductive neural network learning and a neural network version of explanation-based learning. The latter component analyzes games using knowledge that was previously learned from expert play. Particular care has been taken in the design of an appropriate feature representation, sampling methods, and parameter settings. Thus far, NeuroChess has successfully managed to beat GNU-Chess in several hundreds of games. However, the level of play still compares poorly to GNU-Chess and human chess players.

Despite the initial success, NeuroChess faces two fundamental problems which both might well be in the way of excellent chess play. Firstly, training time is limited, and it is to be expected that excellent chess skills develop only with excessive training time. This is particularly the case if only the final outcomes are considered. Secondly, with each step of TD-learning NeuroChess loses information. This is partially because the features used for describing chess boards are incomplete, *i.e.*, knowledge about the feature values alone does not suffice to determine the actual board exactly. But, more importantly, neural networks have not the discriminative power to assign arbitrary values to all possible feature combinations. It is therefore unclear that a TD-like approach will ever, for example, develop good chess openings.

Another problem of the present implementation is related to the trade-off between knowledge and search. It has been well recognized that the ultimate cost in chess is determined by the time it takes to generate a move. Chess programs can generally invest their time in search, or in the evaluation of chess boards (search-knowledge trade-off) [3]. Currently, NeuroChess does a poor job, because it spends most of its time computing board evaluations. Computing a large neural network function takes two orders of magnitude longer than evaluating an optimized linear evaluation function (like that of GNU-Chess). VLSI neural network technology offers a promising perspective to overcome this critical shortcoming of sequential neural network simulations.

## Acknowledgment

The author gratefully acknowledges the guidance and advise by Hans Berliner, who provided the features for representing chess boards, and without whom the current level of play would be much worse. He also thanks Tom Mitchell for his suggestion on the learning methods, and Horst Aurisch for his help with GNU-Chess and the database.

## Footnotes

[1]This is because in the current version NeuroChess still heavily uses expert games for sampling. Whenever a grand-master moves its queen to the center of the board, the queen is usually safe, and there is indeed a positive correlation between having the queen in the center and winning in the database. NeuroChess falsely deduces that having the queen in the center is good. This effect disappears when the level of self-play is increased, but this comes at the expense of drastically increased training time, since self-play requires search.

## References

[1] Thomas S. Anantharaman. *A Statistical Study of Selective Min-Max Search in Computer Chess.* PhD thesis, Carnegie Mellon University, School of Computer Science, Pittsburgh, PA, 1990. Technical Report CMU-CS-90-173.

[2] R. E. Bellman. *Dynamic Programming.* Princeton University Press, Princeton, NJ, 1957.

[3] Hans J. Berliner, Gordon Goetsch, Murray S. Campbell, and Carl Ebeling. Measuring the performance potential of chess programs. *Artificial Intelligence,* 43:7–20, 1990.

[4] Justin A. Boyan. Generalization in reinforcement learning: Safely approximating the value function. In G. Tesauro, D. Touretzky, and T. Leen, editors, *Advances in Neural Information Processing Systems 7,* San Mateo, CA, 1995. Morgan Kaufmann. (to appear).

[5] Gerald DeJong and Raymond Mooney. Explanation-based learning: An alternative view. *Machine Learning,* 1(2):145–176, 1986.

[6] Michael Gherrity. *A Game-Learning Machine.* PhD thesis, University of California, San Diego, 1993.

[7] Tom M. Mitchell, Rich Keller, and Smadar Kedar-Cabelli. Explanation-based generalization: A unifying view. *Machine Learning,* 1(1):47–80, 1986.

[8] Tom M. Mitchell and Sebastian Thrun. Explanation based learning: A comparison of symbolic and neural network approaches. In Paul E. Utgoff, editor, *Proceedings of the Tenth International Conference on Machine Learning,* pages 197–204, San Mateo, CA, 1993. Morgan Kaufmann.

[9] Tom M. Mitchell and Sebastian Thrun. Explanation-based neural network learning for robot control. In S. J. Hanson, J. Cowan, and C. L. Giles, editors, *Advances in Neural Information Processing Systems 5,* pages 287–294, San Mateo, CA, 1993. Morgan Kaufmann.

[10] A. L. Samuel. Some studies in machine learning using the game of checkers. *IBM Journal on research and development,* 3:210–229, 1959.

[11] Johannes Schäfer. Erfolgsorientiertes Lernen mit Tiefensuche in Bauernendspielen. Technical report, Universität Karlsruhe, 1993. (in German).

[12] Nikolaus Schraudolph, Pater Dayan, and Terrence J. Sejnowski. Using the TD(lambda) algorithm to learn an evaluation function for the game of go. In *Advances in Neural Information Processing Systems 6,* San Mateo, CA, 1994. Morgan Kaufmann.

[13] Patrice Simard, Bernard Victorri, Yann LeCun, and John Denker. Tangent prop – a formalism for specifying selected invariances in an adaptive network. In J. E. Moody, S. J. Hanson, and R. P. Lippmann, editors, *Advances in Neural Information Processing Systems 4,* pages 895–903, San Mateo, CA, 1992. Morgan Kaufmann.

[14] Richard S. Sutton. Learning to predict by the methods of temporal differences. *Machine Learning,* 3, 1988.

[15] Prasad Tadepalli. Planning in games using approximately learned macros. In *Proceedings of the Sixth International Workshop on Machine Learning,* pages 221–223, Ithaca, NY, 1989. Morgan Kaufmann.

[16] Gerald J. Tesauro. Practical issues in temporal difference learning. *Machine Learning,* 8, 1992.

[17] Sebastian Thrun and Anton Schwartz. Issues in using function approximation for reinforcement learning. In M. Mozer, P. Smolensky, D. Touretzky, J. Elman, and A. Weigend, editors, *Proceedings of the 1993 Connectionist Models Summer School,* Hillsdale, NJ, 1993. Erlbaum Associates.